# Hyperparameter Learning for Graph Based Semi-supervised Learning Algorithms

**Xinhua Zhang**[*]
Statistical Machine Learning Program
National ICT Australia, Canberra, Australia
and CSL, RSISE, ANU, Canberra, Australia
xinhua.zhang@nicta.com.au

**Wee Sun Lee**
Department of Computer Science
National University of Singapore
3 Science Drive 2, Singapore 117543
leews@comp.nus.edu.sg

## Abstract

Semi-supervised learning algorithms have been successfully applied in many applications with scarce labeled data, by utilizing the unlabeled data. One important category is graph based semi-supervised learning algorithms, for which the performance depends considerably on the quality of the graph, or its hyperparameters. In this paper, we deal with the less explored problem of learning the graphs. We propose a graph learning method for the harmonic energy minimization method; this is done by minimizing the leave-one-out prediction error on labeled data points. We use a gradient based method and designed an efficient algorithm which significantly accelerates the calculation of the gradient by applying the matrix inversion lemma and using careful pre-computation. Experimental results show that the graph learning method is effective in improving the performance of the classification algorithm.

## 1  Introduction

Recently, graph based semi-supervised learning algorithms have been used successfully in various machine learning problems including classification, regression, ranking, and dimensionality reduction. These methods create graphs whose vertices correspond to the labeled and unlabeled data while the edge weights encode the similarity between each pair of data points. Classification is performed using these graphs by labeling unlabeled data in such a way that instances connected by large weights are given similar labels. Example graph based semi-supervised algorithms include min-cut [3], harmonic energy minimization [11], and spectral graphical transducer [8].

The performance of the classifier depends considerably on the similarity measure of the graph, which is normally defined in two steps. Firstly, the weights are defined *locally* in a pair-wise parametric form using functions that are essentially based on a distance metric such as radial basis functions (RBF). It is argued in [7] that modeling error can degrade performance of semi-supervised learning. As the distance metric is an important part of graph based semi-supervised learning, it is crucial to use a good distance metric. In the second step, smoothing is applied *globally*, typically, based on the spectral transformation of the graph Laplacian [6, 10].

There have been only a few existing approaches which address the problem of graph learning. [13] learns a nonparametric spectral transformation of the graph Laplacian, assuming that the weight and distance metric are given. [9] learns the spectral parameters by performing evidence maximization using approximate inference and gradient descent. [12] uses evidence maximization and Laplace approximation to learn simple parameters of the similarity function. Instead of learning one single good graph, [4] proposed building robust graphs by applying random perturbation and edge removal

---
[*]This work was done when the author was at the National University of Singapore.

from an ensemble of minimum spanning trees. [1] combined graph Laplacians to learn a graph. Closest to our work is [11], which learns different bandwidths for different dimensions by minimizing the entropy on unlabeled data; like the maximum margin motivation in transductive SVM, the aim here is to get confident labeling of the data by the algorithm.

In this paper, we propose a new algorithm to learn the hyperparameters of distance metric, or more specifically, the bandwidth for different dimensions in the RBF form. In essence, these bandwidths are just model parameters and normal model selection methods include *k*-fold cross validation or leave-one-out (LOO) cross validation in the extreme case can be used for selecting the bandwidths. Motivated by the same spirit, we base our learning algorithm on the aim of achieving low LOO prediction loss on labeled data, i.e., each labeled data can be correctly classified by the other labeled data in a semi-supervised style with as high probability as possible. This idea is similar to [5] which learns multiple parameters for SVM. Since most LOO style algorithms are plagued with prohibitive computational cost, an efficient algorithm is designed. With a simple regularizer, the experimental results show that learning the hyperparameters by minimizing the LOO loss is effective.

## 2   Graph Based Semi-supervised Learning

Suppose we have a set of labeled data points $\{(x_i, y_i)\}$ for $i \in L \triangleq \{1, ..., l\}$. In this paper, we only consider binary classification, i.e., $y_i \in \{1 \text{ (positive)}, 0 \text{ (negative)}\}$. In addition, we also have a set of unlabeled data points $\{x_i\}$ for $i \in U \triangleq \{l+1, ..., l+u\}$. Denote $n \triangleq l + u$. Suppose the dimensionality of input feature vectors is $m$.

### 2.1   Graph Based Classification Algorithms

One of the earliest graph based semi-supervised learning algorithms is *min-cut* by [3], which minimizes:

$$E(f) \triangleq \sum_{i,j} w_{ij}(f_i - f_j)^2 \tag{1}$$

where the nonnegative $w_{ij}$ encodes the similarity between instance $i$ and $j$. The label $f_i$ is fixed to $y_i \in \{1, 0\}$ if $i \in L$. The optimization variables $f_i$ $(i \in U)$ are constrained to $\{1, 0\}$. This combinatorial optimization problem can be efficiently solved by the max-flow algorithm. [11] relaxed the constraint $f_i \in \{1, 0\}$ $(i \in U)$ to real numbers. The optimal solution of the unlabeled data's soft labels can be written neatly as:

$$f_U = (D_U - W_{UU})^{-1}W_{UL}f_L = (I - P_{UU})^{-1}P_{UL}f_L \tag{2}$$

where $f_L$ is the vector of soft labels (fixed to $y_i$) for $L$. $D \triangleq diag(d_i)$, where $d_i \triangleq \sum_j w_{ij}$ and $D_U$ is the submatrix of $D$ associated with unlabeled data. $P \triangleq D^{-1}W$. $W_{UU}, W_{UL}, P_{UU}$, and $P_{UL}$ are defined by:

$$W = \begin{pmatrix} W_{LL} & W_{LU} \\ W_{UL} & W_{UU} \end{pmatrix}, P = \begin{pmatrix} P_{LL} & P_{LU} \\ P_{UL} & P_{UU} \end{pmatrix}.$$

The solution (2) has a number of interesting properties pointed out by [11]. All $f_i$ $(i \in U)$ are automatically bounded by $[0, 1]$, so it is also known as square interpolation. They can be interpreted by using Markov random walk on the graph. Imagine a graph with $n$ nodes corresponding to the $n$ data points. Define the probability of transferring from $x_i$ to $x_j$ as $p_{ij}$, which is actually row-wise normalization of $w_{ij}$. The random walk starts from any unlabeled points, and stops once it hits any labeled point (absorbing boundary). Then $f_i$ is the probability of hitting a positive labeled point. In this sense, the labeling of each unlabeled point is largely based on its neighboring labeled points, which helps to alleviate the problem of noisy data. (1) can also be interpreted as a quadratic energy function and its minimizer is known to be harmonic: $f_i$ $(i \in U)$ equals the average of $f_j$ $(j \neq i)$ weighted by $p_{ij}$. So we call this algorithm Harmonic Energy Minimization (HEM). By (1), $f_U$ is independent of $w_{ii}$ $(i = 1, ..., n)$, so henceforth we fix $w_{ii} = p_{ii} = 0$.

Finally, to translate the soft labels $f_i$ to hard labels pos/neg, the simplest way is by thresholding at 0.5, which works well when the two classes are well separated. [11] proposed another approach, called Class Mass Normalization (CMN), to make use of prior information such as class ratio in unlabeled data, estimated by that in labeled data. Specifically, they normalize the soft labels to $f_i^+ \triangleq$ $f_i \Big/ \sum_{j=1}^n f_j$ as the probabilistic score of being positive, and to $f_i^- \triangleq (1 - f_i) \Big/ \sum_{j=1}^n (1 - f_j)$ as

the score of being negative. Suppose there are $r_+$ positive points and $r_-$ negative points in the labeled data, then we classify $x_i$ to positive iff $f_i^+ r_+ > f_i^- r_-$.

## 2.2 Basic Hyperparameter Learning Algorithms

One of the simplest parametric form of $w_{ij}$ is RBF:

$$w_{ij} = \exp\left(-\sum_d (x_{i,d} - x_{j,d})^2 / \sigma_d^2\right) \tag{3}$$

where $x_{i,d}$ is the $d^{th}$ component of $x_i$, and likewise the meaning of $f_{U,i}$ in (4). The bandwidth $\sigma_d$ has considerable influence on the classification accuracy. HEM uses one common bandwidth for all dimensions, which can be easily selected by cross validation. However, it will be desirable to learn different $\sigma_d$ for different dimensions; this allows a form of feature selection. [11] proposed learning the hyperparameters $\sigma_d$ by minimizing the entropy on unlabeled data points (we call it MinEnt):

$$H(f_U) = -\sum_{i=1}^{u} (f_{U,i} \log f_{U,i} + (1 - f_{U,i}) \log(1 - f_{U,i})) \tag{4}$$

The optimization is conducted by gradient descent. To prevent numerical problems, they replaced $P$ with $\tilde{P} = \epsilon \mathcal{U} + (1 - \epsilon)P$, where $\epsilon \in [0, 1)$, and $\mathcal{U}$ is the uniform matrix with $\mathcal{U}_{ij} = n^{-1}$.

# 3 Leave-one-out Hyperparameter Learning

In this section, we present the formulation and efficient calculation of our graph learning algorithm.

## 3.1 Formulation and Efficient Calculation

We propose a graph learning algorithm which is similar to minimizing the leave-one-out cross validation error. Suppose we hold out a labeled example $x_t$ and predict its label by using the rest of the labeled and unlabeled examples. Making use of the result in (2), the soft label for $x_t$ is $s^\top f_U^t$ (the first component of $f_U^t$), where

$$s \triangleq (1, 0, ..., 0)^\top \in \mathbb{R}^{u+1} , \, f_U^t \triangleq (f_0^t, f_{l+1}^t, ..., f_n^t)^\top.$$

Here, the value of $f_U^t$ can be determined by $f_U^t = (I - \tilde{P}_{UU}^t)^{-1} \tilde{P}_{UL}^t f_L^t$, where

$$f_L^t \triangleq (f_1, .., f_{t-1}, f_{t+1}, ..., f_l)^\top, \, \tilde{p}_{ij} \triangleq (1 - \varepsilon)p_{ij} + \varepsilon/n , \, P_{UU}^t \triangleq \begin{pmatrix} p_{tt} & p_{tU} \\ p_{Ut} & P_{UU} \end{pmatrix},$$

$$p_{Ut} \triangleq (p_{l+1,t}, ..., p_{n,t})^\top , \, p_{tU} \triangleq (p_{t,l+1}, ..., p_{t,n}) ,$$

$$P_{UL}^t \triangleq \begin{pmatrix} p_{t,1} & \cdots & p_{t,t-1} & p_{t,t+1} & \cdots & p_{t,l} \\ p_{l+1,1} & \cdots & p_{l+1,t-1} & p_{l+1,t+1} & \cdots & p_{l+1,l} \\ \cdots & \cdots & \cdots & \cdots & \cdots & \cdots \\ p_{n,1} & \cdots & p_{n,t-1} & p_{n,t+1} & \cdots & p_{n,l} \end{pmatrix}.$$

If $x_t$ is positive, then we hope that $f_{U,1}^t$ is as close to 1 as possible. Otherwise, if $x_t$ is negative, we hope that $f_{U,1}^t$ is as close to 0 as possible. So the cost function to be minimized can be written as:

$$Q = \sum_{t=1}^{l} h_t(f_{U,1}^t) = \sum_{t=1}^{l} h_t\left(s^\top (I - \tilde{P}_{UU}^t)^{-1} \tilde{P}_{UL}^t f_L^t\right) \tag{5}$$

where $h_t(x)$ is the cost function for instance $t$. We denote $h_t(x) = h^+(x)$ for $y_t = 1$ and $h_t(x) = h^-(x)$ for $y_t = 0$. Possible choices of $h^+(x)$ include $1 - x$, $(1-x)^a$, $a^{-x}$, and $-\log x$ with $a > 1$. Possible choices for $h^-(x)$ include $x$, $x^a$, $a^{x-1}$, and $-\log(1-x)$. Let $Loo\_loss(x_t, y_t) \triangleq h_t(f_{U,1}^t)$.

To minimize $Q$, we use gradient-based optimization methods. The gradient is:

$$\partial Q / \partial \sigma_d = \sum_{t=1}^{l} h_t'(f_{U,1}^t) s^\top (I - \tilde{P}_{UU}^t)^{-1} \left(\partial \tilde{P}_{UU}^t / \partial \sigma_d \cdot f_U^t + \partial \tilde{P}_{UL}^t / \partial \sigma_d \cdot f_L^t\right),$$

using matrix property $d\boldsymbol{X}^{-1} = -\boldsymbol{X}^{-1}(d\boldsymbol{X})\boldsymbol{X}^{-1}$. Denoting $(\beta^t)^\top \triangleq h_t'(f_{U,1}^t) s^\top (I - \tilde{P}_{UU}^t)^{-1}$ and noting $\tilde{P} = \varepsilon \mathcal{U} + (1 - \varepsilon)P$, we have

$$\partial Q / \partial \sigma_d = (1 - \varepsilon) \sum_{t=1}^{l} (\beta^t)^\top \left(\partial P_{UU}^t / \partial \sigma_d \cdot f_U^t + \partial P_{UL}^t / \partial \sigma_d \cdot f_L^t\right). \tag{6}$$

Since in both $P_{UU}^t$ and $P_{UL}^t$, the first row corresponds to $x_t$, and the $i^{th}$ row ($i \geq 2$) corresponds to $x_{i+l-1}$, denoting $P_{UN}^t \triangleq (P_{UL}^t \, P_{UU}^t)$ makes sense as each row of $P_{UN}^t$ corresponds to a well

defined single data point. Let all notations about $P$ carry over to the corresponding $W$. We now use $sw_i^t \triangleq \sum_{k=1}^n w_{UN}^t(i,k)$ and $\sum_{k=1}^n \partial w_{UN}^t(i,k)/\partial\sigma_d$ ($i = 1, ..., u+1$) to denote the sum of these corresponding rows. Now (6) can be rewritten in ground terms by the following "two" equations:

$$\partial P_{U\bullet}^t(i,j)/\partial\sigma_d = (sw_i^t)^{-1}\left(\partial w_{U\bullet}^t(i,j)/\partial\sigma_d - p_{U\bullet}^t(i,j)\sum_{k=1}^n \partial w_{UN}^t(i,k)/\partial\sigma_d\right),$$

where $\bullet$ can be $U$ or $L$. $\partial w_{ij}/\partial\sigma_d = 2w_{ij}(x_{i,d} - x_{j,d})^2/\sigma_d^3$ by (3).

The naïve way to calculate the function value $Q$ and its gradient is presented in Algorithm 1. We call it leave-one-out hyperparameter learning (LOOHL).

---

**Algorithm 1** naïve form of LOOHL

---
1: function value $Q \leftarrow 0$, gradient $g \leftarrow (0, ..., 0)^\top \in \mathbb{R}^m$
2: **for** each $t = 1, ..., l$ (leave-one-out loop for each labeled point) **do**
3:     $f_L^t \leftarrow (f_1, .., f_{t-1}, f_{t+1}, ..., f_l)^\top,$          $f_U^t \leftarrow (I - \tilde{P}_{UU}^t)^{-1}\tilde{P}_{UL}^t f_L^t,$
        $Q \leftarrow Q + h_t\left(f_{U,1}^t\right),$                     $(\beta^t)^\top \leftarrow h_t'(f_{U,1}^t)s^\top(I - \tilde{P}_{UU}^t)^{-1}$
4:     **for** each $d = 1, ..., m$ (for all feature dimensions) **do**
5:         $\frac{\partial P_{UU}^t(i,j)}{\partial\sigma_d} \leftarrow \frac{1}{sw_i^t}\left(\frac{\partial w_{UU}^t(i,j)}{\partial\sigma_d} - p_{UU}^t(i,j)\sum_{k=1}^n \frac{\partial w_{UN}^t(i,k)}{\partial\sigma_d}\right)$
                                      where $sw_i^t = \sum_{k=1}^n w_{UN}^t(i,k), \quad i,j = 1, ..., u+1$
6:         $\frac{\partial P_{UL}^t(i,j)}{\partial\sigma_d} \leftarrow \frac{1}{sw_i^t}\left(\frac{\partial w_{UL}^t(i,j)}{\partial\sigma_d} - p_{UL}^t(i,j)\sum_{k=1}^n \frac{\partial w_{UN}^t(i,k)}{\partial\sigma_d}\right)$     $i = 1, ..., u+1, j = 1, ..., l-1$
7:         $g_d \leftarrow g_d + (1 - \epsilon)(\beta^t)^\top\left(\frac{\partial P_{UU}^t}{\partial\sigma_d}f_U^t + \frac{\partial P_{UL}^t}{\partial\sigma_d}f_L^t\right)$
8:     **end for**
9: **end for**

---

The computational complexity of the naïve algorithm is expensive: $O(lu(mn+u^2))$, just to calculate the gradient once. Here we assume the cost of inverting a $u \times u$ matrix is $O(u^3)$. We reduce the two terms in the cost by means of using matrix inversion lemma and careful pre-computation.

One part of the cost, $O(lu^3)$, stems from inverting $I - \tilde{P}_{UU}^t$, a $(u+1) \times (u+1)$ matrix, for $l$ times in (5). We note that for different $t$, $I - \tilde{P}_{UU}^t$ differs only by the first row and first column. So there exist two vectors $\alpha, \beta \in \mathbb{R}^{u+1}$ such that $I - \tilde{P}_{UU}^{t_1} = (I - \tilde{P}_{UU}^{t_2}) + e\alpha^\top + \beta e^\top$, where $e = (1, 0, ..., 0)^\top \in \mathbb{R}^{u+1}$. With $I - \tilde{P}_{UU}^t$ expressed in this form, we are ready to apply matrix inversion lemma:

$$\left(A + \alpha\beta^\top\right)^{-1} = A^{-1} - A^{-1}\alpha \cdot \beta^\top A^{-1}/\left(1 + \alpha^\top A\beta\right). \tag{7}$$

We only need to invert $I - \tilde{P}_{UU}^t$ for $t = 1$ from scratch, and then apply (7) twice for each $t \geqslant 2$. The new total complexity related to matrix inversion is $O\left(u^3 + lu^2\right)$.

The other part of the cost, $O(lumn)$, can be reduced by using careful pre-computation. Written in detail, we have:

$$\frac{\partial Q}{\partial\sigma_d} = \sum_{t=1}^l \sum_{i=1}^{u+1} \frac{\beta_i^t}{sw_i^t}\left(\sum_{j=1}^{u+1} \frac{\partial w_{UU}^t(i,j)}{\partial\sigma_d}f_{U,j}^t + \sum_{j=1}^{l-1} \frac{\partial w_{UL}^t(i,j)}{\partial\sigma_d}f_{L,j}^t\right.$$

$$\left. - \sum_{k=1}^n \frac{\partial w_{UN}^t(i,k)}{\partial\sigma_d}\left(\sum_{j=1}^{u+1} p_{UU}^t(i,j) f_{U,j}^t + \sum_{j=1}^{l-1} p_{UL}^t(i,j) f_{L,j}^t\right)\right) \triangleq \sum_{i=1}^n \sum_{j=1}^n \alpha_{ij}\frac{\partial w_{ij}}{\partial\sigma_d}$$

The crucial observation is the existence of $\alpha_{ij}$, which are independent of dimension index $d$. Therefore, they can be pre-computed efficiently. The Algorithm 2 below presents the efficient approach to gradient calculation.

---

**Algorithm 2** Efficient algorithm to gradient calculation

---
1: **for** $i, j = 1, ..., n$ **do**
2:     **for** all feature dimension $d$ on which either $x_i$ or $x_j$ is nonzero **do**
3:         $g_d = g_d + \alpha_{ij} \cdot \partial w_{ij}/\partial\sigma_d$
4:     **end for**
5: **end for**

---

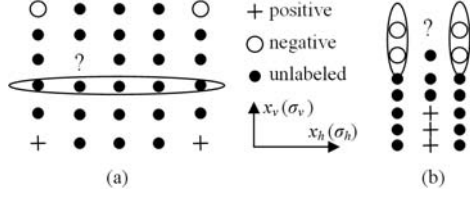

Figure 1: Examples of degenerative graphs learned by pure LOOHL.

Letting $sw_i \triangleq \sum_{k=1}^{n} w_{ik}$ and $\delta(\cdot)$ be *Kroneker* delta, we derive the form of $\alpha_{ij}$ as:

$$\alpha_{ij} = sw_i^{-1} \sum_{t=1}^{l} \beta_{i-l+1}^{t} \left( f_{U,j-l+1}^{t} - \sum_{k=l+1}^{n} p_{ik} f_{U,k-l+1}^{t} - p_{it} f_{U,1}^{t} - \sum_{k=1:k\neq t}^{l} p_{ik} f_{k} \right) \text{ for } i > l \text{ and } j > l;$$

$$\alpha_{ij} = sw_i^{-1} \sum_{t=1}^{l} \beta_{i-l+1}^{t} \left( f_{U,1}^{t} \delta(t=j) + f_{j} \delta(t \neq j) - p_{it} f_{U,1}^{t} - \sum_{k=l+1}^{n} p_{ik} f_{U,k-l+1}^{t} - \sum_{k=1:k\neq t}^{l} p_{ik} f_{k} \right)$$
$$\text{for } i > l \text{ and } j \leqslant l;$$

$$\alpha_{ij} = sw_i^{-1} \beta_1^i \left( f_{U,j-l+1}^{i} - \sum_{k=l+1}^{n} p_{ik} f_{U,k-l+1}^{i} - \sum_{k=1}^{l} p_{ik} f_{k} \right) \qquad \text{for } i \leqslant l \text{ and } j > l;$$

$$\alpha_{ij} = sw_i^{-1} \beta_1^i \left( f_{j} - \sum_{k=l+1}^{n} p_{ik} f_{U,k-l+1}^{i} - \sum_{k=1}^{l} p_{ik} f_{k} \right) \qquad \text{for } i \leqslant l \text{ and } j \leqslant l,$$

and $\alpha_{ii}$ are fixed to 0 for all $i$ since we fix $w_{ii} = p_{ii} = 0$.

All $\alpha_{ij}$ can be computed in $O(u^2 l)$ time and Algorithm 2 can be completed in $O(n^2 \tilde{m})$ time, where

$$\tilde{m} \triangleq 2n^{-1}(n-1)^{-1} \cdot \sum_{1 \leqslant i < j \leqslant n} |\{d \in 1...m| \ x_i \text{ or } x_j \text{ is not zero on feature } d\}|.$$

In many applications such as text classification and image pattern recognition, the data is very sparse and $\tilde{m} \ll m$. In sum, the computational cost has been reduced from $O(lu(mn + u^2))$ to $O(lnu + n^2\tilde{m} + u^3)$. The space cost is mild at $O(n^2 + n\tilde{m})$.

## 4  Regularizing the Graph Learning

Similar to the MinEnt method, purely applying LOOHL can lead to degenerative graphs. In this section, we show two such examples and then propose a simple approach which regularizes the graph learning process.

Two degenerative graphs are shown in Figure 1. In example (a), the points with the same $x_v$ coordinate are from the same classes. For each labeled point, there is another labeled point from the opposite class which has the same $x_h$ coordinate. So the leave-one-out hyperparameter learning will push $1/\sigma_h$ to zero and $1/\sigma_v$ to infinity, i.e., all points can transfer only horizontally. Therefore the graph will effectively split into six disconnected sub-graphs, each sharing the same $x_v$ coordinate as showed in (a). So the desired gradual change of label from positive to negative along dimension $x_v$ cannot appear. As a result, the point at question mark cannot hit any labeled points and cannot be classified. One way to prevent such degenerate graphs is to prevent $1/\sigma_v$ from growing too large, e.g., with a regularizer such as $\sum_d (1/\sigma_d)^2$.

In example (b), although the negative points will encourage both horizontal and vertical walk, horizontal walk will make the leave-one-out error large on positive points. So the learned $1/\sigma_v$ will be far smaller than $1/\sigma_h$, i.e., the result strongly encourages walking in vertical direction and ignoring the information from the horizontal direction. As a result, the point at the question mark will be labeled as positive, although by nearest neighbor intuition, it should be labeled as negative. We notice that the four negative points will be partitioned into two groups as shown in the figure. In such a case, the regularizer $\sum_d (1/\sigma_d)^2$ will not be helpful with utilizing dimensions that are informative. A different regularizer that encourages the use of more dimensions may be better in this case. One simple regularizer that has this property is to minimize the variance of the inverse bandwidth $\sum_d (1/\sigma_d - \mu)^2$, where $\mu = m^{-1} \sum_d 1/\sigma_d$, assuming that the mean is non-zero. It

Table 1: Dataset properties. Sparsity is the average frequency of features to be zero in the whole dataset. The rightmost column gives the size of the whole dataset from which the labeled data in experiment is sampled. Some data in text dataset has unknown label, thus always used as unlabeled.

| dataset name | feature property: continuous or discrete (how many levels), sparsity | | Number of features | | dataset size (#positive: #negative: #unlabelled) |
|---|---|---|---|---|---|
| | Original | Probe | Original | Probe | |
| handwritten digits (4 vs 9) | 256 levels, 79.7% | 1000 levels, 85.8% | 475 | 4525 | 1000 : 1000 : 0 |
| cancer vs normal | continuous, 50.3% | 1000 levels, 45.4% | 18543 | 9961 | 88 : 112 : 0 |
| Reuters text categorization | continuous, 99.0% | 1000 levels, 99.3% | 4608 | 13459 | 300 : 300 : 1400 |
| compounds bind to thrombin | binary, 99.43% | binary, 98.55% | 31976 | 58417 | 112 : 1038 : 0 |
| Ionosphere | continuous, 38.3% | | 33 | | 225 : 126: 0 |

is a priori unclear which regularizer will be better empirically, but for the datasets in our experiments, the minimum variance regularizer is overwhelmingly better, even when useless features are intentionally added to the datasets.

Since the gradient based optimization can get stuck in local minima, it is advantageous to test several different parameter initialization. With this in mind, we implement a simple approximation to the minimum variance regularizer that tests different parameter initialization as well. We discretize $\mu$ and minimize the leave-one-out loss plus $\sum_d \left(1/\sigma_d - 1/\tilde{\sigma}\right)^2$, where $\tilde{\sigma}$ is fixed a priori to several different possible values. We run with different $\tilde{\sigma}$ and set all initial $\sigma_d$ to $\tilde{\sigma}$. Then we choose the function produced by the value of $\tilde{\sigma}$ that has the smallest regularized cost function value. This process is similar to restarting from various values to avoid local minima, but now we are also trying with different mean of estimated optimal bandwidth at the same time. A similar way to regularize is by using a Gaussian prior with mean $\mu^{-1}$ and minimizing $Q + C \sum_d \left(1/\sigma_d - 1/\mu\right)^2$ with respect to $\sigma_d$ and $\mu$ simultaneously.

## 5   Experimental Results

Using HEM as a basis for classification, we compare the test accuracy of three model selection methods: LOOHL, 5-CV (tying all bandwidths and choose by 5-fold cross validation), and MinEnt, each with both thresholding and CMN. Since the topic of this paper is how to learn the hyperparameters of a graph, we pay more attention to how the performance of a *given* recognized classifier can be improved by means of learning the graph, than to the comparison between different classifers' performance, i.e., comparing with other semi-supervised or supervised learning algorithms. Ionosphere is from UCI repository. The other four datasets used in the experiment are from NIPS 2003 Workshop on feature selection challenge. Each of them has two versions: original version and probe version which adds useless probing features in order to investigate the algorithm's performance in the presence of useless features, though at current stage we do not use the algorithm as feature selector. Since the workshop did not provide the original datasets, we downloaded the original datasets from other sites. Our original intention was to use original versions that we downloaded and to reproduce the probe version ourself using the pre-processing described in NIPS 2003 workshop, so that we can check the performance of the algorithms on datasets with and without redundant features. Unfortunately, we find that with our own effort at pre-processing, the datasets with probes yield far different accuracies compared with the datasets with probes downloaded from the workshop web site. Thus we are using the original version and the probe version downloaded from difference sources, and the comparison between them should be done with care, though the demonstration of LOOHL's efficacy is not affected.

The properties of the five datasets are summarized in Table 1. We randomly pick the labeled subset $L$ from all labeled data available under the constraint that both classes must be present in $L$. The remaining labeled and unlabeled data are used as unlabeled data. For example, by saying $|L| = 20$ for *text* dataset, we mean randomly picking 20 points from the 600 labeled data as labeled, and label the other 1980 points by using our algorithm. Finally we calculate the prediction accuracy on the 580 (originally) labeled points. For other datasets, say *cancer*, testing is on 180 points since we know the label of all points. For each fixed $|L|$, this random test is conducted for 10 times and the average accuracy is reported. Then $|L|$ is varied. We normalized all input feature vectors to have length 1.

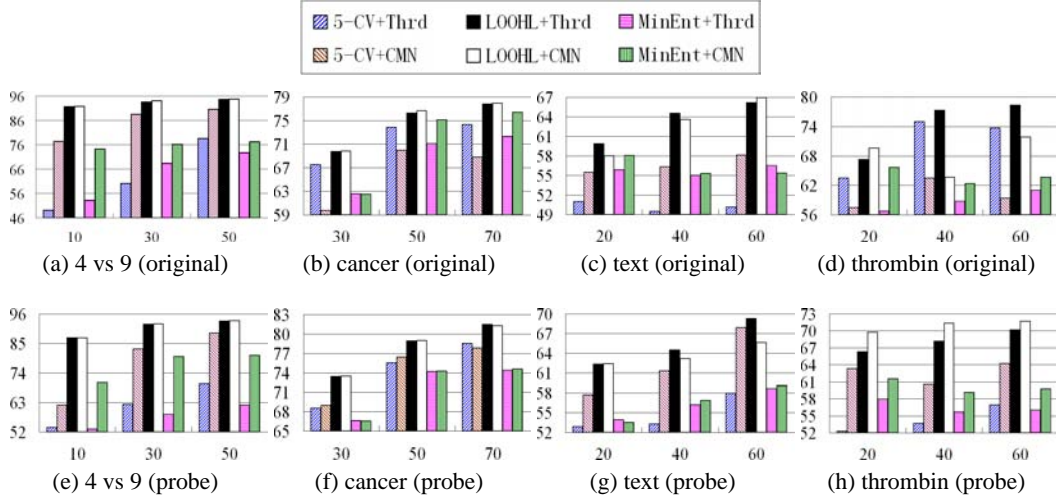

Figure 2: Accuracy of original and probe versions in percentage vs. number of labeled data.

The initial common bandwidth and smoothing factor $\epsilon$ in MinEnt are selected by five fold cross validation. For LOOHL, We fix $h^+(x) = (1-x)^2$ and $h^-(x) = x^2$. The final objective function is:

$$C_1 \times Loo\_loss\_Normal + C_2 \times \sum_d (1/\sigma_d - 1/\tilde{\sigma})^2 \big/ m,$$

$$Loo\_loss\_Normal \triangleq (2r_+)^{-1} \sum_{y_i=1} Loo\_loss(x_i, y_i) + (2r_-)^{-1} \sum_{y_i=0} Loo\_loss(x_i, y_i), \quad (8)$$

and there are $r_+$ positive labeled examples and $r_-$ negative labeled examples. For each $C_1$:$C_2$ ratio, we run on $\tilde{\sigma} = 0.05, 0.1, 0.15, 0.2, 0.25, 0.3$ for all datasets and select the function that corresponds to the smallest objective function value for use in cross validation testing. The final $C_1$:$C_2$ value was picked by five fold cross validation, with discrete levels at $10^{-i}$, where $i = 1, 2, 3, 4, 5$, since strong regularizer is needed given the large number of features (variables) and much fewer labeled points. The optimization solver we use is the Toolkit for Advanced Optimization [2].

From the results in Figure 2 and Figure 3, we can make the following observations and conclusions:

1. LOOHL generally outperforms 5-CV and MinEnt. Both LOOHL+Thrd and LOOHL+CMN outperform 5-CV and MinEnt (regardless of Thrd or CMN) on all datasets except thrombin and ionosphere, where either LOOHL+CMN or LOOHL+Thrd finally performs best.

2. For 5-CV, CMN is almost always better than thresholding, except on the original form of cancer and thrombin dataset, where CMN hurts 5-CV. In [11], it is claimed that although the theory of HEM is sound, CMN is still necessary to achieve reasonable performance because the underlying graph is often poorly estimated and may not reflect the classification goal, i.e., one should not rely exclusively on the graph. Now that our LOOHL is aimed at learning a good graph, the ideal case is that the graph learned is suitable for our classification such that the improvement by CMN will not be large. In other words, the difference between LOOHL+CMN and LOOHL+Thrd, compared with the difference between 5-CV+CMN and 5-CV+Thrd, can be viewed as an approximate indicator of how well the graph is learned by LOOHL.

The efficacy of LOOHL can be clearly observed in datasets 4vs9, cancer, text, ionosphere and original version of thrombin. In these cases, we see that LOOHL+Thrd is already achieving high accuracy and LOOHL+CMN does not offer much improvement then or even hurts performance due to inaccurate class ratio estimation. In fact, LOOHL+Thrd performs reliably well on all datasets. It is thus desirable to learn the bandwidth for each dimension of the feature vector, and there is no longer any need to post-process by using class ratio information.

3. The performance of MinEnt is generally inferior to 5-CV and LOOHL. MinEnt+Thrd has equal chance of out-performing or losing to 5-CV+Thrd, while 5-CV+CMN is almost always better than MinEnt+CMN. Most of the time, MinEnt+CMN performs significantly better than MinEnt+Thrd, so we can conclude that MinEnt fails to learn a good graph. This may be due to converging to a poor local minimum, or that the idea of minimizing the entropy on unlabeled data is by itself insufficient.

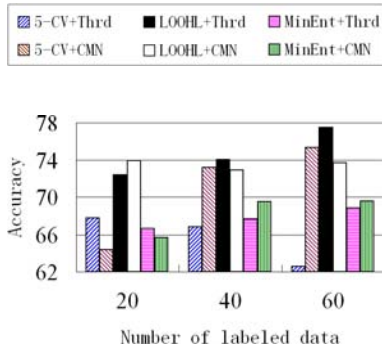

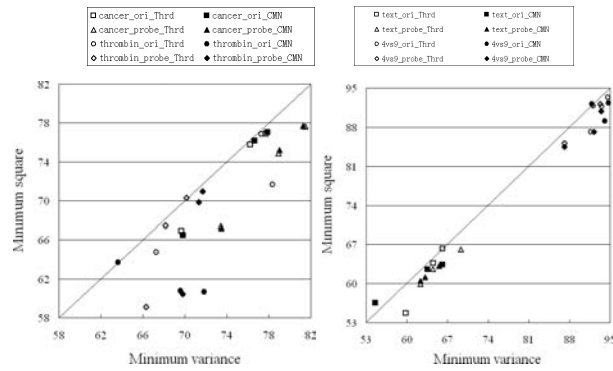

Figure 3: Accuracy of Ionosphere in percentage vs. number of labeled data.

Figure 4: Accuracy comparison of priors in percentage between minimizing sum of square inverse bandwidth $\sum_d \sigma_d^{-2}$ and minimizing variance of inverse bandwidth.

4. For these datasets, assuming low variance of inverse bandwidth with discretization as regularizer is more reasonable than assuming that many features are irrelevant to the classification. This is even true for probe versions of the datasets. Figure 4 shows the comparison.

# 6 Conclusions

In this paper, we proposed learning the graph for graph based semi-supervised learning by minimizing the leave-one-out prediction error, with a simple regularizer. Efficient gradient calculation algorithms are designed and the empirical result is encouraging.

### Acknowledgements

This work is partially funded by the Singapore-MIT Alliance. National ICT Australia is funded through the Australian Government's *Backing Australia's Ability* initiative, in part through the Australian Research Council.

## References

[1] Andreas Argyriou, Mark Herbster, and Massimiliano Pontil. Combining Graph Laplacians for Semi-Supervised Learning. In *NIPS 2005*, Vancouver, Canada, 2005.

[2] Steven Benson, Lois McInnes, Jorge Moré, and Jason Sarich. TAO User Manual ANL/MCS-TM-242, http://www.mcs.anl.gov/tao, 2005.

[3] Avrin Blum, and Shuchi Chawla. Learning From Labeled and Unlabeled Data using Graph Mincuts. In *ICML 2001*.

[4] Miguel Á Carreira-Perpiñán, and Richard S. Zemel. Proximity Graphs for Clustering and Manifold Learning. In *NIPS 2004*.

[5] Olivier Chapelle, Vladimir Vapnik, Olivier Bousquet, and Sayan Mukherjee. Choosing Multiple Parameters for Support Vector Machines. *Machine Learning*, 46, 131–159, 2002.

[6] Olivier Chapelle, Jason Weston, and Bernhard Schölkopf. Cluster Kernels for Semi-Supervised Learning. In *NIPS 2002*.

[7] Fabio G. Cozman, Ira Cohen, and Marcelo C. Cirelo. Semi-Supervised Learning of Mixture Models and Bayesian Networks. In *ICML 2003*.

[8] Thorsten Joachims. Transductive Learning via Spectral Graph Partitioning. In *ICML 2003*.

[9] Ashish Kapoor, Yuan Qi, Hyungil Ahn, and Rosalind Picard. Hyperparameter and Kernel Learning for Graph Based Semi-Supervised Classification. In *NIPS 2005*.

[10] Alexander Smola, and Risi Kondor. Kernels and Regularization on Graphs. In *COLT 2003*.

[11] Xiaojin Zhu, Zoubin Ghahramani, and John Lafferty. Semi-Supervised Learning Using Gaussian Fields and Harmonic Functions. In *ICML 2003*.

[12] Xiaojin Zhu, John Lafferty, and Zoubin Ghahramani. Semi-Supervised Learning: From Gaussian Fields to Gaussian Processes. CMU Technical Report CMU-CS-03-175.

[13] Xiaojin Zhu, Jaz Kandola, Zoubin Ghahramani, and John Lafferty. Non-parametric Transforms of Graph Kernels for Semi-Supervised Learning. In *NIPS 2004*.
